# Adaptive Classification by Variational Kalman Filtering

**Peter Sykacek**
Department of Engineering Science
University of Oxford
Oxford, OX1 3PJ, UK
*psyk@robots.ox.ac.uk*

**Stephen Roberts**
Department of Engineering Science
University of Oxford
Oxford, OX1 3PJ, UK
*sjrob@robots.ox.ac.uk*

## Abstract

We propose in this paper a probabilistic approach for adaptive inference of generalized nonlinear classification that combines the computational advantage of a parametric solution with the flexibility of sequential sampling techniques. We regard the parameters of the classifier as latent states in a first order Markov process and propose an algorithm which can be regarded as variational generalization of standard Kalman filtering. The variational Kalman filter is based on two novel lower bounds that enable us to use a non-degenerate distribution over the adaptation rate. An extensive empirical evaluation demonstrates that the proposed method is capable of infering competitive classifiers both in stationary and non-stationary environments. Although we focus on classification, the algorithm is easily extended to other generalized nonlinear models.

## 1 Introduction

The demand for adaptive learning methods, e.g. for use in brain computer interfaces (BCIs) [15] has recently triggered a considerable interest in such algorithms. We may approach adaptive learning with algorithms that were designed for stationary environments and use *learning rates* to make these methods adaptive. These approaches can be traced back to early work on learning algorithms (e.g. [1]). A more recent account to this approach is [17], who combines the probabilistic method of sequential variational inference ([9]) and a forgetting factor to obtain an adaptive learning method. Probabilistic or *Bayesian* methods allow also for a completely different interpretation of adaptive learning. We may regard the model coefficients as *latent* (i.e. unobserved) states of a first order Markov process.

$$p(\boldsymbol{w}_{n-1}|\mathcal{D}_{n-1}) \tag{1}$$
$$p(\boldsymbol{w}_n|\boldsymbol{w}_{n-1}, \lambda\boldsymbol{I})$$
$$p(y_n|\boldsymbol{x}_n, \boldsymbol{w}_n)$$

The posterior distribution, $p(\boldsymbol{w}_{n-1}|\mathcal{D}_{n-1})$, at state $n-1$ summarizes all information obtained about the model. This posterior and the conditional distribution, $p(\boldsymbol{w}_n|\boldsymbol{w}_{n-1}, \lambda\boldsymbol{I})$, represent the prior for the following state. The conditional distribution can be thought of as additive *process* or *state* noise with precision $\lambda$. Predictions are obtained by a probabilistic observation model $p(y_n|\boldsymbol{x}_n, \boldsymbol{w}_n)$. Using this model, we obtain an appropriate adaptation

rate by hierarchical Bayesian inference of the process noise precision $\lambda$. Equation (1) suggests that we may interpret adaptive Bayesian inference as generalization of the well known Kalman filter ([12]). This view of adaptive learning has been used by [6], who use extended Kalman filtering to obtain a *Laplace* approximation of the posterior over $\boldsymbol{w}_n$ and maximum likelihood II ([3]) for inference of the adaptation rate. Another generalization of Kalman filtering are the recently quite popular particle filters (e.g. [7]). Being Monte Carlo methods, particle filters have over Laplace approximations the advantage of much greater flexibility. This comes however at the expense of a higher representational and computational complexity. To combine the flexibility of particle filtering with the computational advantage of parametric methods, we propose a variational approximation (e.g. [11] , [2] and [8]) for inference of the Markov process in Equation (1). Unlike maximum likelihood II, the variational Kalman filter allows us to have a *non* degenerate distribution over the process noise precision. We derive in this paper a variational Kalman filter classifier and show with an extensive empirical evaluation that the resulting classifiers obtain excellent generalization accuracies both in stationary and non-stationary domains.

## 2 Methods

### 2.1 A generalized nonlinear classifier

Classification is a prediction problem, where some *regressor*, $\boldsymbol{x}_n$, predicts the *expectation* of a *response variable* $y_n$. Since a $k$ categorical polytomous solution is easily recovered from $k-1$ dichotomous solutions ([16], pages 44-45), we restrict all further discussions to dichotomos classification using $0/1$ responses. We thus have only one degree of freedom and predict the binary probability, $P(y_n \equiv 0|\boldsymbol{w}, \boldsymbol{x}_n)$, which depends on the model parameters $\boldsymbol{w}$. To obtain a flexible discriminant, we use a generalized nonlinear model, i.e. a radial basis function (RBF) network ([14] and [5]), with logistic output transformation (Equation (3)).

$$\boldsymbol{\phi}_n \quad = \quad [1, \boldsymbol{x}_n, \boldsymbol{\varphi}(\boldsymbol{x}_n; \boldsymbol{w}_\varphi)]^T \ , \eta_n = \boldsymbol{\phi}_n^T \boldsymbol{w} \tag{2}$$

$$P(y_n|\boldsymbol{w}, \boldsymbol{w}_\varphi, \boldsymbol{x}_n) \quad = \quad (1 + \exp((2y_n - 1)\eta_n))^{-1} \tag{3}$$

The classifier has a nonlinear feature space $\boldsymbol{\phi}_n$ which for reasons of adaptivity depends on $\boldsymbol{w}_\varphi$ and a linear mapping into latent space $\eta_n$. We allow for Gaussian basis functions, i.e. $\varphi_k(\boldsymbol{x}_n) = \exp(-0.5\kappa_k(\boldsymbol{x}_n^T\boldsymbol{\mu}_k)^2)$ or thin plate splines, i.e. $\varphi_k(\boldsymbol{x}_n) = |\boldsymbol{x}_n^T\boldsymbol{\mu}_k| \log(|\boldsymbol{x}_n^T\boldsymbol{\mu}_k|)$. Both basis functions are parameterized by their center locations $\boldsymbol{\mu}_k$. Since we want to have a simple unimodal posterior over model parameters, we update the coefficients of the basis set $\boldsymbol{w}_\varphi$ randomly according to a Metropolis Hastings kernel ([13]) and solve for the conditional posterior $p(\boldsymbol{w}|\boldsymbol{w}_\varphi, \mathcal{D}_n)$ analytically.

### 2.2 The variational Kalman filter

In order to ease discussion of adaptive inference, we illustrate the dependencies implied by Equation (1) in figure 1 as a directed acyclic graph (DAG). In accordance with Kalman filtering, we assume a Gaussian posterior at time $n-1$ with mean $\hat{\boldsymbol{w}}_{n-1}$ and precision $\boldsymbol{\Lambda}_{n-1}$ and zero mean Gaussian state noise with isotropic precision $\lambda\boldsymbol{I}$. Inference of $\lambda$ is based on a "flat" proper Gamma prior specified by parameters $\alpha$ and $\beta$. In order to obtain reasonable posteriors over $\lambda$, we follow [10] and assume constant adaptation within a window of size $N$. The proposed variational Bayesian approach ignores the anti-causal information flow and is thus based on maximizing a lower bound on the logarithmic model evidence of a windowed Kalman filter. Following these assumptions, we obtain the expression for the log evidence in Equation (4) by substituting the generalized nonlinear model (Equations (2) to (3)) into the formulation of adaptive Bayesian learning (1). We have then to make all

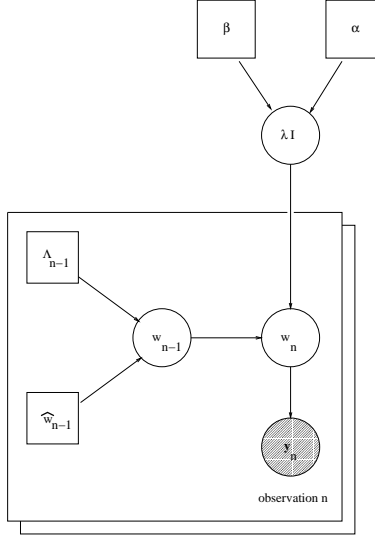

Figure 1: This figure illustrates adaptive inference as a directed acyclic graph. The coefficients of the classifier, $\boldsymbol{w}_n$, are assumed to be Gaussian, following a first order Markov process. The hyper parameter $\lambda$ is given a Gamma prior specified by parameters $\alpha$ and $\beta$.

distributions explicit and integrate over all model coefficients, which is done analytically over all prior states $\boldsymbol{w}_{n-1}$.

$$
\begin{aligned}
\log(p(\mathcal{D}_N)) &= \log\Big(\int_\lambda \prod_{n=1}^{N}\Big[\int_{\boldsymbol{w}_n}(2\pi)^{-\frac{d}{2}}|\boldsymbol{\Lambda}_{n-1}^{-1}+\lambda^{-1}\boldsymbol{I}|^{-\frac{1}{2}} \\
&\times \exp(-0.5(\boldsymbol{w}_n-\hat{\boldsymbol{w}}_{n-1})^T(\boldsymbol{\Lambda}_{n-1}^{-1}+\lambda^{-1}\boldsymbol{I})^{-1}(\boldsymbol{w}_n-\hat{\boldsymbol{w}}_{n-1})) \\
&\times (1+\exp((2y_n-1)\boldsymbol{\phi}_n^T\boldsymbol{w}_n))^{-1}d\boldsymbol{w}_n\Big] \\
&\times \frac{\beta^\alpha}{\Gamma(\alpha)}\lambda^{(\alpha-1)}\exp(-\beta\lambda)d\lambda\Big).
\end{aligned}
\tag{4}
$$

The structure of Equation (4) suggests that the approximate posterior $Q(\boldsymbol{w}_n)$ can be chosen to be Gaussian and the approximate posterior $Q(\lambda)$ can be chosen to be a Gamma distribution. These functional forms do however not simply result from a mean field approximation of the posterior as $Q(\lambda)\prod_{n=1}^{N}Q(\boldsymbol{w}_n)$. In order to obtain the required conjugacy, we have to use lower bounds for the probability of the target label, $(1+\exp((2y_n-1)\boldsymbol{\phi}_n^T\boldsymbol{w}_n))^{-1}$ and for both $|\boldsymbol{\Lambda}_{n-1}^{-1}+\lambda^{-1}\boldsymbol{I}|^{-\frac{1}{2}}$ and $\exp(-0.5(\boldsymbol{w}_n-\hat{\boldsymbol{w}}_{n-1})^T(\boldsymbol{\Lambda}_{n-1}^{-1}+\lambda^{-1}\boldsymbol{I})^{-1}(\boldsymbol{w}_n-\hat{\boldsymbol{w}}_{n-1}))$.

### 2.3 Variational lower bounds

In order to achieve conjugacy with a Gaussian distribution, we use the lower bound for the logistic sigmoid proposed in [9]

$$
\begin{aligned}
\log(P(y_n|\boldsymbol{\phi}_n,\boldsymbol{w}_n)) &\geq -\frac{(2y_n-1)\boldsymbol{\phi}_n^T\boldsymbol{w}_n}{2}-\log(2)-\log\left(\cosh\left(\frac{\xi_n}{2}\right)\right) \\
&- \frac{\tanh(\frac{\xi_n}{2})}{4\xi_n}\left(\left(\frac{\boldsymbol{\phi}_n^T\boldsymbol{w}_n}{2}\right)^2-\xi_n^2\right),
\end{aligned}
\tag{5}
$$

in which $\xi_n$ are the variational parameters of a locally linear expansion in $((2y_n - 1)\phi_n^T w_n)^2$ of every prediction contained in the window. In order to get expressions that are conjugate with a Gamma distribution over the process noise precision $\lambda$, we derive two novel lower bounds. Assuming a $d$-dimensional parameter vector $w_n$, we get

$$-0.5 \log |\Lambda_{n-1}^{-1} + \lambda^{-1} I| \geq \frac{d}{2} \log \lambda - \frac{1}{2} \log |\nu \Lambda_n^{-1} + I| \qquad (6)$$
$$- \frac{1}{2}(\lambda - \nu)\mathrm{tr}(\nu I + \Lambda_n)^{-1}$$

and

$$-0.5(w_n - \hat{w}_{n-1})^T (\Lambda_{n-1}^{-1} + \lambda^{-1} I)^{-1}(w_n - \hat{w}_{n-1}) \geq \qquad (7)$$
$$-0.5(w_n - \hat{w}_{n-1})^T (\Lambda_{n-1}^{-1} + \nu^{-1} I)^{-1}(w_n - \hat{w}_{n-1})$$
$$-0.5(\lambda - \nu)(w_n - \hat{w}_{n-1})^T (\nu \Lambda_{n-1}^{-1} + I)^{-2}(w_n - \hat{w}_{n-1}),$$

which are expressions in $\lambda$ and $\log(\lambda)$ and thus conjugate with a Gamma distribution. Both bounds are expanded in the *identical* parameter $\nu$ which is justified since both are linear expansions in $(\lambda - \nu)$ and maximization must thus lead to identical values. Using these lower bounds together with a mean field assumption, $Q(\lambda) \prod_{n=1}^{N} Q(w_n)$, and the usual Jensens inequalities, we immediately obtain a *negative free energy* as lower bound of the log evidence in Equation (4). For reasons of brevity we do not include this expression here.

## 2.4 Parameter updates

In order to distinguish between the parameters of the prior and posterior distributions, we henceforth denote the latter with superscript $Q$. Inference requires to maximize the negative free energy with respect to all variational parameters. These are the coefficients of the $N$ Gaussian distributions, $Q(w_n)$, the $N$ parameters in the bounds of the logistic sigmoid, $\xi_n$, the coefficients of the Gamma posterior over the noise process precision, $Q(\lambda)$ and the parameter in the Gamma conjugacy bounds, $\nu$. Maximization with respect to $Q(w_n)$ results in a Gaussian distribution with precision $\Lambda_n^Q$ and mean $\hat{w}_n^Q$.

$$\Lambda_n^Q = (\Lambda_{n-1}^{-1} + \nu^{-1} I)^{-1} + \frac{\tanh(\frac{\xi_n}{2})}{2\xi_n} \phi_n \phi_n^T \qquad (8)$$

$$\hat{w}_n^Q = (\Lambda_n^Q)^{-1} \left( (\Lambda_{n-1}^{-1} + \nu^{-1} I)^{-1} \hat{w}_{n-1} + \frac{2y_n - 1}{2} \phi_n \right)$$

Maximization with respect to $Q(\lambda)$ results in a Gamma distribution with location parameter $\alpha^Q$ and scale parameter $\beta^Q$.

$$\alpha^Q = \alpha + \frac{Nd}{2} \qquad (9)$$

$$\beta^Q = \beta + \frac{1}{2} \sum_{n=1}^{N} \Big[ \mathrm{tr}(\nu I + \Lambda_{n-1})^{-1} + (\hat{w}_n^Q - \hat{w}_{n-1})^T (\nu \Lambda_{n-1}^{-1} + I)^{-2}$$
$$\times (\hat{w}_n^Q - \hat{w}_{n-1}) + \mathrm{tr}((\Lambda_n^Q)^{-1}(\nu \Lambda_{n-1}^{-1} + I)^{-2}) \Big]$$

According to [9], maximization with respect to $\xi_n$ leads to

$$\xi_n = \sqrt{(\phi_n^T \hat{w}_n^Q)^2 + \phi_n^T \Lambda_n^Q \phi_n}. \qquad (10)$$

Maximization with respect to the variational parameter $\nu$ leads for both bounds to

$$\nu = \frac{\alpha^Q}{\beta^Q}. \qquad (11)$$

In order to allow the basis mapping in Equation (2) to track modifications in the input data distributions, we propose the perturbation $\boldsymbol{w}'_\varphi = \boldsymbol{w}_\varphi + \boldsymbol{\delta}$, where $\boldsymbol{\delta} \sim N(\mathbf{0}, \boldsymbol{\Delta})$ is drawn from a Gaussian and accept the proposal according to probability

$$P_{acc} = \min \left( 1, \frac{\exp\left( \sum_{n=1}^{N} \frac{2y_n - 1}{2} \boldsymbol{\phi}'_n - \frac{\tanh(\frac{\xi_n}{2})}{4\xi_n} \boldsymbol{\phi}'^T_n (\hat{\boldsymbol{w}}^Q_n \hat{\boldsymbol{w}}^{QT}_n + \boldsymbol{\Lambda}^{Q-1}_n) \boldsymbol{\phi}'_n \right)}{\exp\left( \sum_{n=1}^{N} \frac{2y_n - 1}{2} \boldsymbol{\phi}_n - \frac{\tanh(\frac{\xi_n}{2})}{4\xi_n} \boldsymbol{\phi}^T_n (\hat{\boldsymbol{w}}^Q_n \hat{\boldsymbol{w}}^{QT}_n + \boldsymbol{\Lambda}^{Q-1}_n) \boldsymbol{\phi}_n \right)} \right) \quad (12)$$

If we assume that the negative free energy describes the log evidence exactly, this is a Metropolis Hastings kernel ([13]) that leaves the marginal posterior $p(\boldsymbol{w}_\varphi | \mathcal{D}_n)$ invariant. We could thus represent the marginal posterior with random samples. For computational reasons however, we use the scheme only for random updates of $\boldsymbol{w}_\varphi$. An algorithm for parameter inference will first propose a random update of $\boldsymbol{w}_\varphi$ and then iterate maximizations according to Equation (8) to Equation (11) until we observe convergence of the negative free energy. Alternatively we can use a fixed number of iterations, for which our experiments suggest that 15 iterations suffice.

## 2.5 Model predictions

Since we do not know the response when predicting, we have to sum the negative free energy over $y_n$. This results in a new expression for $\hat{\boldsymbol{w}}^Q_n$ which we obtain from Equation (8) by dropping the term that depends on $y_n$. Due to the dependency on $\xi_n$, maximization with respect to $Q(\boldsymbol{w}_n)$ has to alternate with maximization with respect to $\xi_n$, the latter again being done according to Equation (10). Having reached convergence, we obtain an approximate log probability for $y_n$ by taking the expectation of the bound of the sigmoid in Equation (5) with respect to $Q(\boldsymbol{w}_n)$ and maximizing with respect to $\xi_n$.

$$\log(\tilde{P}(y_n | \boldsymbol{\phi}_n)) = -\frac{(2y_n - 1)\boldsymbol{\phi}^T_n \boldsymbol{w}_n}{2} - \log(2) - \log\left( \cosh\left( \frac{\xi_n}{2} \right) \right). \quad (13)$$

Exponentiating the approximate log probabilities results in a *sub probability* measure over $y_n$ with $\sum_{y_n} \tilde{P}(y_n | \boldsymbol{\phi}_n) \leq 1$, with the difference $1 - \sum_{y_n} \tilde{P}(y_n | \boldsymbol{\phi}_n)$ representing an additional uncertainty about $y_n$, introduced by the approximation of the logistic sigmoid.

## 3 Experiments

All experiments reported in this section use a model with 10 Gaussian basis functions with precision $\kappa_k = 0.25$. For updating the basis, we use zero mean Gaussian random variates with precision $\boldsymbol{\Delta} = 1000\boldsymbol{I}$. The initial prior over parameters is a zero mean Gaussian with isotropic precision $\boldsymbol{\Lambda}_0 = 0.1\boldsymbol{I}$. For maximizing the negative free energy we use 15 iterations. The first experiment aims at obtaining a parametrization for $\alpha$, $\beta$ and the window length, $N$, that allows us to make inferences of the process noise $\lambda$ that are insensitive to the actual "drift" of the problem. We use for that purpose the test set from the synthetic problem in [16][1]. The samples of this balanced problem are reshuffled such that consecutive class labels differ. In order to get a non-stationarity, we swap the class labels in the second half of the data. The results shown in figure 2 are obtained with $\alpha = 0.01$ and $\beta = 10^{-4}$. We propose these settings together with a window size $N = 10$, because this is a good compromise between fast tracking and high stationary accuracy.

We are now ready to compare the algorithm with an equivalent static classifier using several public data sets and classification of single trial EEG which, due to learning effects in humans, is known to be non-stationary. In order to avoid that the model has an influence on

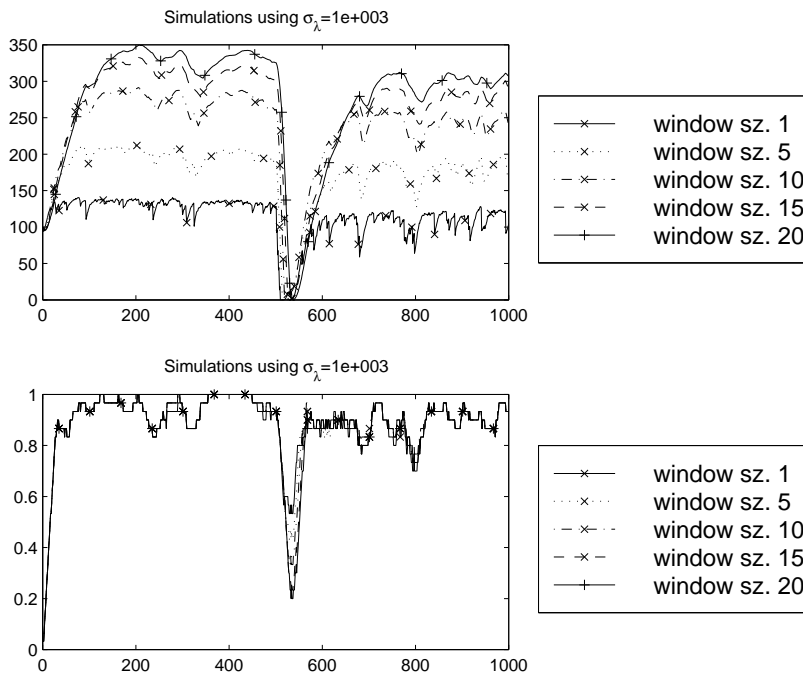

Figure 2: Results obtained on Ripleys' synthetic data set with swapped class labels after sample 500. The top graph shows the expected value of the precision of the noise process, $< \lambda >_{Q(\lambda)} = \frac{\alpha^Q}{\beta^Q}$ for different window sizes (i.e. for different numbers of samples used for infering the adaptation rate). The bottom graph shows the instantaneous generalization accuracy estimated in a window of size 30. The prior over $\lambda$ is a Gamma distribution with expectation 100 and variance $10^6$.

the results, we compare the generalization accuracy of the variational Kalman filter classi-fier (vkf) with an identical non-adaptive model. Inference of the static model is based on sequential variational learning ([9]). We obtain sequential variational inference (svi) from our approach by setting $\lambda$ in Equation (1) to infinity. The comparisons are evaluated for significance using McNemar's test, a method for analyzing paired results that is suggested in [16]. The comparison uses vehicle data[2], satellite image data, Johns Hopkins University ionosphere data, balance scale weight and distance data and the wine recognition database, all taken from the StatLog database which is available at the UCI repository ([4]). The satellite image data set is used as is provided with 4435 samples in the training and 2000 samples in the test set. Vehicle data are merged such that we have 500 samples in the train-ing and 252 in the test set. The other data were split into two equal sized data sets, which were both used as training and independent test sets respectively. We also use the pima diabetes data set from [16][3]. Table 1 compares the generalization accuracies (in fractions) obtained with the variational Kalman filter with generalization accuracies obtained with sequential variational inference. The probability of the null hypothesis, $P_{null}$, that both classifiers are equal suggests that only the differences for the Balance scale and the Pima Indian data sets are significant, with either method being better in one case. Since the gen-eralization accuracies of both methods are almost identical, we conclude that if applied to

| Data sets | Generalization results | | |
|---|---|---|---|
| | vkf | svi | $P_{null}$ |
| J.H.U. ionosphere | 0.87 | 0.88 | 0.41 |
| Satellite image | 0.81 | 0.81 | 0.29 |
| Balance scale | 0.89 | 0.87 | 0.03 |
| Pima diabetes | 0.76 | 0.80 | 0.03 |
| Vehicle | 0.77 | 0.77 | 0.42 |
| Wine | 0.97 | 0.95 | 0.25 |

Table 1: Generalization accuracies obtained with the variational Kalman filter (vkf) and sequential variational inference (svi).

| Cognitive task | Generalization results | | |
|---|---|---|---|
| | vkf | svi | $P_{null}$ |
| rest/move, no feedback | 0.69 | 0.61 | 0.00 |
| rest/move, feedback | 0.71 | 0.70 | 0.39 |
| move/math, no feedback | 0.69 | 0.62 | 0.00 |
| move/math, feedback | 0.64 | 0.60 | 0.00 |

Table 2: Generalization accuracies obtained for classification of single trial EEG show that the variational Kalman filter significantly improves the results in three out of four cases.

stationary problems, we may expect the variational Kalman filter to obtain generalization accuracies that are similar to those of static methods.

In order to assess the variational Kalman filter on a non-stationary problem, we apply it to classification of single trial EEG, a problem which is part of BCIs. The data for this experiment has been obtained from eight untrained subjects that perform two different task combinations (rest EEG vs. imagined movements and imagined movements vs. a mathematical task), once without and once with visual feedback. For one cognitive experiment each pair of tasks is repeated ten times. We classify on a one second basis an thus have per subject and task combination 200 samples. The regressors in this experiment are three reflection coefficients (a parametrization of autoregressive models, see e.g. [18]). The comparison in table 2 reports within subject results obtained by two fold cross testing. Using half of the data, we allow for convergence of the methods before estimating the generalization accuracy on the other half of the data. The generalization accuracies in table 2 are averaged across subjects. We obtain in three out of four experiments a significant improvement with the variational Kalman filter.

## 4 Discussion

We propose in this paper a parametric approach for adaptive inference of nonlinear classification. Our algorithm can be regarded as variational generalization of Kalman filtering which we obtain by using two novel lower bounds that allow us to have a non-degenerate distribution over the adaptation rate. Inference is done by iteratively maximizing a lower bound of the log evidence. As a result we obtain an approximate posterior that is a product of a multivariate Gaussian and a Gamma distribution. Our simulations have shown that the approach is capable of infering classifiers that have good generalization performance both in stationary and non-stationary domains. In situations with moderate sized latent spaces, e.g. in the BCI experiments reported above, prediction and parameter updates can be done in real time on conventional PCs. Although we focus on classification, the algorithm is

based on general ideas and thus easily applicable to other generalized nonlinear models.

## Acknowledgements

We would like to express gratitude to the anonymous reviewers of this paper for their valuable suggestions for improving the paper. Peter Sykacek is currently supported by grant Nr. F46/399 kindly provided by the BUPA foundation.

## Footnotes

[1]This data set can be obtained at http://www.stats.ox.ac.uk/pub/PRNN/.

[2]Vehicle data was donated to StatLog by the Turing Institute Glasgow, Scotland.

[3]This data set can be obtained at http://www.stats.ox.ac.uk/pub/PRNN/.

## References

[1] S.-I. Amari. A theory of adaptive pattern classifiers. *IEEE Transactions on Electronic Computers*, 16:299–307, 1967.

[2] H. Attias. Inferring parameters and structure of latent variable models by variational Bayes. In *Proc. 15th Conf. on Uncertainty in AI, 1999*, 1999.

[3] J. O. Berger. *Statistical Decision Theory and Bayesian Analysis*. Springer, New York, 1985.

[4] C.L. Blake and C.J. Merz. UCI repository of machine learning databases. http://www.ics.uci.edu/~mlearn/MLRepository.html, 1998. University of California, Irvine, Dept. of Information and Computer Sciences.

[5] D. S. Broomhead and D. Lowe. Multivariable functional interpolation and adaptive networks. *Complex Systems*, 2:321–355, 1988.

[6] J.F.G. de Freitas, M. Niranjan, and A.H. Gee. Regularisation in Sequential Learning Algorithms. In M. Jordan, M. Kearns, and S. Solla, editors, *Advances in Neural Information Processing Systems (NIPS 10)*, pages 458–464, 1998.

[7] A. Doucet, J. F. G. de Freitas, and N. Gordon, editors. *Sequential Monte Carlo Methods in Practice*. Springer-Verlag, 2001.

[8] Z. Ghahramani and M. J. Beal. Variational inference for Bayesian mixture of factor analysers. In *Advances in Neural Information Processing Systems 12*, pages 449–455, 2000.

[9] T. S. Jaakkola and M. I. Jordan. Bayesian parameter estimation via variational methods. *Statistics and Computing*, 10:25–37, 2000.

[10] A.H. Jazwinski. Adaptive filtering. *Automatica*, pages 475–485, 1969.

[11] M. I. Jordan, Z. Ghahramani, T. S. Jaakkola, and L. K. Saul. An introduction to variational methods for graphical models. In M. I. Jordan, editor, *Learning in Graphical Models*. MIT Press, Cambridge, MA, 1999.

[12] R. E. Kalman. A new approach to linear filtering and prediction problems. *Trans. ASME, J. Basic Eng.*, 82:35–45, 1960.

[13] N. Metropolis, A. Rosenbluth, M. Rosenbluth, A. Teller, and E. Teller. Equations of state calculations by fast computing machines. *Journal of Chemical Physics*, 21:1087–1091, 1953.

[14] J. Moody and C. J. Darken. Fast learning in networks of locally-tuned processing units. *Neural Computation*, 1:281–294, 1989.

[15] W. Penny, S. Roberts, E. Curran, and M. Stokes. EEG-based communication: a pattern recognition approach. *IEEE Trans. Rehab. Eng.*, pages 214–216, 2000.

[16] B. D. Ripley. *Pattern Recognition and Neural Networks*. Cambridge University Press, Cambridge, 1996.

[17] Masa-aki Sato. Online model selection based on the variational Bayes. *Neural Computation*, pages 1649–1681, 2001.

[18] P. Sykacek and S. Roberts. Bayesian time series classification. In T.G. Dietterich, S. Becker, and Z. Gharamani, editors, *Advances in Neural Processing Systems 14*, pages 937–944. MIT Press, 2002.
